# Stationarity and Stability of Autoregressive Neural Network Processes

Friedrich Leisch[1], Adrian Trapletti[2] & Kurt Hornik[1]

[1] Institut für Statistik
Technische Universität Wien
Wiedner Hauptstraße 8–10 / 1071
A-1040 Wien, Austria
*firstname.lastname*@ci.tuwien.ac.at

[2] Institut für Unternehmensführung
Wirtschaftsuniversität Wien
Augasse 2–6
A-1090 Wien, Austria
adrian.trapletti@wu-wien.ac.at

## Abstract

We analyze the asymptotic behavior of autoregressive neural network (AR-NN) processes using techniques from Markov chains and non-linear time series analysis. It is shown that standard AR-NNs without shortcut connections are asymptotically stationary. If linear shortcut connections are allowed, only the shortcut weights determine whether the overall system is stationary, hence standard conditions for linear AR processes can be used.

## 1 Introduction

In this paper we consider the popular class of nonlinear autoregressive processes driven by additive noise, which are defined by stochastic difference equations of form

$$\xi_t = g(\xi_{t-1}, \ldots, \xi_{t-p}, \theta) + \epsilon_t \qquad (1)$$

where $\epsilon_t$ is an iid. noise process. If $g(\cdots, \theta)$ is a feedforward neural network with parameter ("weight") vector $\theta$, we call Equation 1 an autoregressive neural network process of order $p$, short AR-NN($p$) in the following.

AR-NNs are a natural generalization of the classic linear autoregressive AR($p$) process

$$\xi_t = \alpha_1 \xi_{t-1} + \cdots + \alpha_p \xi_{t-p} + \epsilon_t. \qquad (2)$$

See, e.g., Brockwell & Davis (1987) for a comprehensive introduction into AR and ARMA (autoregressive moving average) models.

One of the most central questions in linear time series theory is the stationarity of the model, i.e., whether the probabilistic structure of the series is constant over time or at least asymptotically constant (when not started in equilibrium). Surprisingly, this question has not gained much interest in the NN literature, especially there are—up to our knowledge—no results giving conditions for the stationarity of AR-NN models. There are results on the stationarity of Hopfield nets (Wang & Sheng, 1996), but these nets cannot be used to estimate conditional expectations for time series prediction.

The rest of this paper is organized as follows: In Section 2 we recall some results from time series analysis and Markov chain theory defining the relationship between a time series and its associated Markov chain. In Section 3 we use these results to establish that standard AR-NN models without shortcut connections are stationary. We also give conditions for AR-NN models with shortcut connections to be stationary. Section 4 examines the NN modeling of an important class of non-stationary time series, namely integrated series. All proofs are deferred to the appendix.

## 2  Some Time Series and Markov Chain Theory

### 2.1  Stationarity

Let $\xi_t$ denote a time series generated by a (possibly nonlinear) autoregressive process as defined in (1). If $\mathbb{E}\epsilon_t = 0$, then $g$ equals the conditional expectation $\mathbb{E}(\xi_t|\xi_{t-1},\ldots,\xi_{t-p})$ and $g(\xi_{t-1},\ldots,\xi_{t-p})$ is the best prediction for $\xi_t$ in the mean square sense.

If we are interested in the long term properties of the series, we may ask whether certain features such as mean or variance change over time or remain constant. The time series is called *weakly stationary* if $\mathbb{E}\xi_t = \mu$ and $cov(\xi_t,\xi_{t+h}) = \gamma_h$, $\forall t$, i.e., mean and covariances do not depend on the time $t$. A stronger criterion is that the whole distribution (and not only mean and covariance) of the process does not depend on the time, in this case the series is called *strictly stationary*. Strong stationarity implies weak stationarity if the second moments of the series exist. For details see standard time series textbooks such as Brockwell & Davis (1987).

If $\xi_t$ is strictly stationary, then $\mathbb{P}(\xi_t \in A) = \pi(A)$, $\forall t$ and $\pi(\cdot)$ is called the *stationary distribution* of the series. Obviously the series can only be stationary from the beginning if it is started with the stationary distribution such that $\xi_0 \sim \pi$. If it is not started with $\pi$, e.g., because $\xi_0$ is a constant, then we call the series *asymptotically stationary* if it converges to its stationary distribution:

$$\lim_{t\to\infty} \mathbb{P}(\xi_t \in A) = \pi(A)$$

### 2.2  Time Series as Markov Chains

Using the notation

$$\begin{aligned} x_{t-1} &= (\xi_{t-1},\ldots,\xi_{t-p})' & (3)\\ G(x_{t-1}) &= (g(x_{t-1}),\xi_{t-1},\ldots,\xi_{t-p+1})' & (4)\\ e_t &= (\epsilon_t,0,\ldots,0)' & (5) \end{aligned}$$

we can write scalar autoregressive models of order $p$ such as (1) or (2) as a first order vector model

$$x_t = G(x_{t-1}) + e_t \qquad (6)$$

with $x_t, e_t \in \mathbb{R}^p$ (e.g., Chan & Tong, 1985). If we write

$$
\begin{aligned}
p^n(x, A) &= \mathbb{P}\{x_{t+n} \in A | x_t = x\} \\
p(x, A) &= p^1(x, A)
\end{aligned}
$$

for the probability of going from point $x$ to set $A \in \mathcal{B}$ in $n$ steps, then $\{x_t\}$ with $p(x, A)$ forms a Markov chain with state space $(\mathbb{R}^p, \mathcal{B}, \lambda)$, where $\mathcal{B}$ are the Borel sets on $\mathbb{R}^p$ and $\lambda$ is the usual Lebesgue measure.

The Markov chain $\{x_t\}$ is called *$\varphi$-irreducible*, if for some $\sigma$-finite measure $\varphi$ on $(\mathbb{R}^p, \mathcal{B}, \lambda)$

$$
\forall x \in \mathbb{R}^p : \quad \sum_{n=1}^{\infty} p^n(x, A) > 0
$$

whenever $\varphi(A) > 0$. This means essentially, that all parts of the state space can be reached by the Markov chain irrespective of the starting point. Another important property of Markov chains is *aperiodicity*, which loosely speaking means that there are no (infinitely often repeated) cycles. See, e.g., Tong (1990) for details.

The Markov chain $\{x_t\}$ is called *geometrically ergodic*, if there exists a probability measure $\pi(A)$ on $(\mathbb{R}^p, \mathcal{B}, \lambda)$ and a $\rho > 1$ such that

$$
\forall x \in \mathbb{R}^p : \quad \lim_{n \to \infty} \rho^n ||p^n(x, \cdot) - \pi(\cdot)|| = 0
$$

where $|| \cdot ||$ denotes the total variation. Then $\pi$ satisfies the invariance equation

$$
\pi(A) = \int p(x, A)\, \pi(dx), \quad \forall A \in \mathcal{B}
$$

There is a close relationship between a time series and its associated Markov chain. If the Markov chain is geometrically ergodic, then its distribution will converge to $\pi$ and the time series is asymptotically stationary. If the time series is started with distribution $\pi$, i.e., $x_0 \sim \pi$, then the series $\{\xi_t\}$ is strictly stationary.

## 3  Stationarity of AR-NN Models

We now apply the concepts defined in Section 2 to the case where $g$ is defined by a neural network. Let $x$ denote a $p$-dimensional input vector, then we consider the following standard network architectures:

**Single hidden layer perceptrons:**

$$
g(x) = \gamma_0 + \sum_i \beta_i \sigma(\alpha_i + a_i' x) \tag{7}
$$

where $\alpha_i$, $\beta_i$ and $\gamma_0$ are scalar weights, $a_i$ are $p$-dimensional weight vectors, and $\sigma(\cdot)$ is a bounded sigmoid function such as $\tanh(\cdot)$.

**Single hidden layer perceptrons with shortcut connections:**

$$
g(x) = \gamma_0 + c'x + \sum_i \beta_i \sigma(\alpha_i + a_i' x) \tag{8}
$$

where $c$ is an additional weight vector for shortcut connections between inputs and output. In this case we define the characteristic polynomial $c(z)$ associated with the linear shortcuts as

$$
c(z) = 1 - c_1 z - c_2 z^2 - \ldots - c^p z^p, \quad z \in \mathbb{C}.
$$

**Radial basis function networks:**

$$g(x) = \gamma_0 + \sum_i \beta_i \phi(\alpha_i |x - m_i|) \tag{9}$$

where $m_i$ are center vectors and $\phi(\cdots)$ is one of the usual bounded radial basis functions such as $\phi(x) = \exp(-x^2)$.

**Lemma 1** *Let $\{x_t\}$ be defined by (6), let $\mathbb{E}|\epsilon_t| < \infty$ and let the PDF of $\epsilon_t$ be positive everywhere in $\mathbb{R}$. Then if $g$ is defined by any of (7), (8) or (9), the Markov chain $\{x_t\}$ is $\phi$-irreducible and aperiodic.*

Lemma 1 basically says that the state space of the Markov chain, i.e., the points that can be reached, cannot be reduced depending on the starting point. An example for a reducible Markov chain would be a series that is always positive if only $x_0 > 0$ (and negative otherwise). This cannot happen in the AR-NN(p) case due to the unbounded additive noise term.

**Theorem 1** *Let $\{\xi_t\}$ be defined by (1), $\{x_t\}$ by (6), further let $\mathbb{E}|\epsilon_t| < \infty$ and the PDF of $\epsilon_t$ be positive everywhere in $\mathbb{R}$. Then*

1.  *If $g$ is a network without linear shortcuts as defined in (7) and (9), then $\{x_t\}$ is geometrically ergodic and $\{\xi_t\}$ is asymptotically stationary.*

2.  *If $g$ is a network with linear shortcuts as defined in (8) and additionally $c(z) \neq 0, \forall z \in \mathbb{C} : |z| \leq 1$, then $\{x_t\}$ is geometrically ergodic and $\{\xi_t\}$ is asymptotically stationary.*

The time series $\{\xi_t\}$ remains stationary if we allow for more than one hidden layer ($\rightarrow$ multi layer perceptron, MLP) or non-linear output units, as long as the overall mapping has bounded range. An MLP with shortcut connections combines a (possibly non-stationary) linear AR(p) process with a non-linear stationary NN part. Thus, the NN part can be used to model non-linear fluctuations around a linear process like a random walk.

The only part of the network that controls whether the overall process is stationary are the linear shortcut connections (if present). If there are no shortcuts, then the process is always stationary. With shortcuts, the usual test for stability of a linear system applies.

## 4 Integrated Models

An important method in classic time series analysis is to first transform a non-stationary series into a stationary one and then model the remainder by a stationary process. The probably most popular models of this kind are autoregressive integrated moving average (ARIMA) models, which can be transformed into stationary ARMA processes by simple differencing.

Let $\Delta^k$ denote the $k$-th order difference operator

$$\Delta \xi_t = \xi_t - \xi_{t-1} \tag{10}$$

$$\Delta^2 \xi_t = \Delta(\xi_t - \xi_{t-1}) = \xi_t - 2\xi_{t-1} + \xi_{t-2} \tag{11}$$

$$\cdots$$

$$\Delta^k \xi_t = \Delta(\Delta \cdots (\Delta \xi_t)) = \sum_{n=0}^{k} (-1)^n \binom{k}{n} \xi_{t-n} \tag{12}$$

with $\Delta^1 = \Delta$. E.g., a standard random walk $\xi_t = \xi_{t-1} + \epsilon_t$ is non-stationary because of the growing variance, but can be transformed into the iid (and hence stationary) noise process $\epsilon_t$ by taking first differences.

If a time series is non-stationary, but can be transformed into a stationary series by taking $k$-th differences, we call the series *integrated of order $k$*. Standard MLPs or RBFs without shortcuts are asymptotically stationary. It is therefore important to take care that these networks are only used to model stationary processes. Of course the network can be trained to mimic a non-stationary process on a finite time interval, but the out-of-sample or prediction performance will be poor, because the network inherently cannot capture some important features of the process. One way to overcome this problem is to first transform the process into a stationary series (e.g., by differencing an integrated series) and train the network on the transformed series (Chng et al., 1996).

As differencing is a linear operation, this transformation can also be easily incorporated into the network by choosing the shortcut connections and weights from input to hidden units accordingly. Assume we want to model an integrated series of integration order $k$, such that

$$\Delta^k \xi_t = g(\Delta^k \xi_{t-1}, \ldots, \Delta^k \xi_{t-p}) + \epsilon_t$$

where $\Delta^k \xi_t$ is stationary. By (12) this is equivalent to

$$
\begin{aligned}
\xi_t &= \sum_{n=1}^{k} (-1)^{n-1} \binom{k}{n} \xi_{t-n} + g(\Delta^k \xi_{t-1}, \ldots, \Delta^k \xi_{t-p}) + \epsilon_t \\
&= \sum_{n=1}^{k} (-1)^{n-1} \binom{k}{n} \xi_{t-n} + \tilde{g}(\xi_{t-1}, \ldots, \xi_{t-p-k}) + \epsilon_t
\end{aligned}
$$

which (for $p > k$) can be modeled by an MLP with shortcut connections as defined by (8) where the shortcut weight vector $c$ is *fixed* to

$$c = \left( \binom{k}{1}, \ldots, (-1)^{p-1} \binom{k}{p} \right)', \qquad \binom{k}{n} := 0 \text{ for } n > k$$

and $\tilde{g}$ is such that $\tilde{g}(\xi_{t-1}, \ldots, \xi_{t-p-k}) = g(\Delta^k x_{t-1})$. This is always possible and can basically be obtained by adding $c$ to all weights between input and first hidden layer of $g$.

An AR-NN(p) can model integrated series up to integration order $p$. If the order of integration is known, the shortcut weights can either be fixed, or the differenced series is used as input. If the order is unknown, we can also train the complete network including the shortcut connections and implicitly estimate the order of integration. After training the final model can be checked for stationarity by looking at the characteristic roots of the polynomial defined by the shortcut connections.

## 4.1 Fractional Integration

Up to now we have only considered integrated series with positive integer order of integration, i.e., $k \in \mathbb{N}$. In the last years models with *fractional* integration order became very popular (again). Series with integration order of $0.5 < k < 1$ can be shown to exhibit self-similar or fractal behavior, and have long memory. These type of processes were introduced by Mandelbrot in a series of paper modeling river flows, e.g., see Mandelbrot & Ness (1968). More recently, self-similar processes were used to model Ethernet traffic by Leland et al. (1994). Also some financial time series such as foreign exchange data series exhibit long memory and self-similarity.

The fractional differencing operator $\Delta^k, k \in [-1, 1]$ is defined by the series expansion

$$\Delta^k \xi_t = \sum_{n=0}^{\infty} \frac{\Gamma(-k+n)}{\Gamma(-k)\Gamma(n+1)} \xi_{t-n} \qquad (13)$$

which is obtained from the Taylor series of $(1 - z)^k$. For $k > 1$ we first use Equation (12) and then the above series for the fractional remainder. For practical computation, the series (13) is of course truncated at some term $n = N$. An AR-NN($p$) model with shortcut connections can approximate the series up to the first $p$ terms.

## 5 Summary

We have shown that AR-NN models using standard NN architectures without shortcuts are asymptotically stationary. If linear shortcuts between inputs and outputs are included—which many popular software packages have already implemented—then only the weights of the shortcut connections determine if the overall system is stationary. It is also possible to model many integrated time series by this kind of networks. The asymptotic behavior of AR-NNs is especially important for parameter estimation, predictions over larger intervals of time, or when using the network to generate artificial time series. Limiting (normal) distributions of parameter estimates are only guaranteed for stationary series. We therefore always recommend to transform a non-stationary series to a stationary series if possible (e.g., by differencing) before training a network on it.

Another important aspect of stationarity is that a single trajectory displays the complete probability law of the process. If we have observed one long enough trajectory of the process we can (in theory) estimate all interesting quantities of the process by averaging over time. This need not be true for non-stationary processes in general, where some quantities may only be estimated by averaging over several independent trajectories. E.g., one might train the network on an available sample and then use the trained network afterwards—driven by artificial noise from a random number generator—to generate new data with similar properties than the training sample. The asymptotic stationarity guarantees that the AR-NN model cannot show "explosive" behavior or growing variance with time.

We currently are working on extensions of this paper in several directions. AR-NN processes can be shown to be strong mixing (the memory of the process vanishes exponentially fast) and have autocorrelations going to zero at an exponential rate. Another question is a thorough analysis of the properties of parameter estimates (weights) and tests for the order of integration. Finally we want to extend the univariate results to the multivariate case with a special interest towards cointegrated processes.

## Acknowledgement

This piece of research was supported by the Austrian Science Foundation (FWF) under grant SFB#010 ('Adaptive Information Systems and Modeling in Economics and Management Science').

# Appendix: Mathematical Proofs

## Proof of Lemma 1

It can easily be shown that $\{x_t\}$ is $\varphi$-irreducible if the support of the probability density function (PDF) of $\epsilon_t$ is the whole real line, i.e., the PDF is positive everywhere in $\mathbb{R}$ (Chan & Tong, 1985). In this case every non-null $p$-dimensional hypercube is reached in $p$ steps with positive probability (and hence every non-null Borel set $A$).

A necessary and sufficient condition for $\{x_t\}$ to be aperiodic is that there exists a set $A$ and positive integer $n$ such that $p^n(x, A) > 0$ and $p^{n+1}(x, A) > 0$ for all $x \in A$ (Tong, 1990, p. 455). In our case this is true for all $n$ due to the unbounded additive noise.

## Proof of Theorem 1

We use the following result from nonlinear time series theory:

**Theorem 2 (Chan & Tong 1985)** *Let $\{x_t\}$ be defined by (1), (6) and let $G$ be compact, i.e. preserve compact sets. If $G$ can be decomposed as $G = G_h + G_d$ and $G_d(\cdot)$ is of bounded range, $G_h(\cdot)$ is continuous and homogeneous, i.e., $G_h(\alpha x) = \alpha G_h(x)$, the origin is a fixed point of $G_h$ and $G_h$ is uniform asymptotically stable, $\mathbb{E}|\epsilon_t| < \infty$ and the PDF of $\epsilon_t$ is positive everywhere in $\mathbb{R}$, then $\{x_t\}$ is geometrically ergodic.*

The noise process $\epsilon_t$ fulfills the conditions by assumption. Clearly all networks are continuous compact functions. Standard MLPs without shortcut connections and RBFs have a bounded range, hence $G_h \equiv 0$ and $G \equiv G_d$, and the series $\{\xi_t\}$ is asymptotically stationary. If we allow for linear shortcut connections between the input and the outputs, we get $G_h = c'x$ and $G_d = \gamma_0 + \sum_i \beta_i \sigma(\alpha_i + a_i'x)$ i.e., $G_h$ is the linear shortcut part of the network, and $G_d$ is a standard MLP without shortcut connections. Clearly, $G_h$ is continuous, homogeneous and has the origin as a fixed point. Hence, the series $\{\xi_t\}$ is asymptotically stationary if $G_h$ is asymptotically stable, i.e., when all characteristic roots of $G_h$ have a magnitude less than unity. Obviously the same is true for RBFs with shortcut connections. Note that the model reduces to a standard linear AR($p$) model if $G_d \equiv 0$.

# References

Brockwell, P. J. & Davis, R. A. (1987). *Time Series: Theory and Methods*. Springer Series in Statistics. New York, USA: Springer Verlag.

Chan, K. S. & Tong, H. (1985). On the use of the deterministic Lyapunov function for the ergodicity of stochastic difference equations. *Advances in Applied Probability*, **17**, 666–678.

Chng, E. S., Chen, S., & Mulgrew, B. (1996). Gradient radial basis function networks for nonlinear and nonstationary time series prediction. *IEEE Transactions on Neural Networks*, **7**(1), 190–194.

Husmeier, D. & Taylor, J. G. (1997). Predicting conditional probability densities of stationary stochastic time series. *Neural Networks*, **10**(3), 479–497.

Jones, D. A. (1978). Nonlinear autoregressive processes. *Proceedings of the Royal Society London A*, **360**, 71–95.

Leland, W. E., Taqqu, M. S., Willinger, W., & Wilson, D. V. (1994). On the self-similar nature of ethernet traffic (extended version). *IEEE/ACM Transactions on Networking*, **2**(1), 1–15.

Mandelbrot, B. B. & Ness, J. W. V. (1968). Fractional brownian motions, fractional noises and applications. *SIAM Review*, **10**(4), 422–437.

Tong, H. (1990). *Non-linear time series: A dynamical system approach*. New York, USA: Oxford University Press.

Wang, T. & Sheng, Z. (1996). Asymptotic stationarity of discrete-time stochastic neural networks. *Neural Networks*, **9**(6), 957–963.
